# A Kernel Subspace Method by Stochastic Realization for Learning Nonlinear Dynamical Systems

**Yoshinobu Kawahara**[*]
Dept. of Aeronautics & Astronautics
The University of Tokyo

**Takehisa Yairi    Kazuo Machida**
Research Center for Advanced Science and Technology
The University of Tokyo

Komaba 4-6-1, Meguro-ku, Tokyo, 153-8904 JAPAN
{kawahara,yairi,machida}@space.rcast.u-tokyo.ac.jp

## Abstract

In this paper, we present a subspace method for learning nonlinear dynamical systems based on stochastic realization, in which state vectors are chosen using kernel canonical correlation analysis, and then state-space systems are identified through regression with the state vectors. We construct the theoretical underpinning and derive a concrete algorithm for nonlinear identification. The obtained algorithm needs no iterative optimization procedure and can be implemented on the basis of fast and reliable numerical schemes. The simulation result shows that our algorithm can express dynamics with a high degree of accuracy.

## 1  Introduction

Learning dynamical systems is an important problem in several fields including engineering, physical science and social science. The objectives encompass a spectrum ranging from the control of target systems to the analysis of dynamic characterization, and for several decades, system identification for acquiring mathematical models from obtained input-output data has been researched in numerous fields, such as system control.

Dynamical systems are learned by, basically, two different approaches. The first approach is based on the principles of minimizing suitable distance functions between data and chosen model classes. Well-known and widely accepted examples of such functions are likelihod functions [1] and the average squared prediction-errors of observed data. For multivariate models, however, this approach is known to have several drawbacks. First, the optimization tends to lead to an ill-conditioned estimation problem because of the over-parameterization, i.e., minimum parameters (called canonical forms) do not exist in multivariate systems. Second, the minimization, except in trivial cases, can only be carried out numerically using iterative algorithms. This often leads to there being no guarantee of reaching a global minimum and high computational costs. The second approach is a subspace method which involves geometric operations on subspaces spanned by the column or row vectors of certain block Hankel matrices formed by input-output data [2,3]. It is well known that subspace methods require no a priori choice of identifiable parameterizations and can be implemented by fast and reliable numerical schemes.

The subspace method has been actively researched throughout the last few decades and several algorithms have been proposed, which are, for representative examples, based on the orthogonal decomposition of input-output data [2,4] and on stochastic realization using canonical correlation analysis [5]. Recently, nonlinear extensions have begun to be discussed for learning systems that cannot be modeled sufficiently with linear expressions. However, the nonlinear algorithms that

---

[*]URL: www.space.rcast.u-tokyo.ac.jp/kawahara/index_e.html

have been proposed to date include only those in which models with specific nonlinearities are assumed [6] or those which need complicated nonlinear regression [7,8]. In this study, we extend the stochastic-realization-based subspace method [5] to the nonlinear regime by developing it on reproducing kernel Hilbert spaces [9], and derive a nonlinear subspace identification algorithm which can be executed by a procedure similar to that in the linear case.

The outline of this paper is as follows. Section 2 gives some theoretical materials for the subspace identification of dynamical systems with reproducing kernels. In section 3, we give some approximations for deriving a practical algorithm, then describe the algorithm specifically in section 4. Finally, an empirical result is presented in section 5, and we give conclusions in section 6.

**Notation**  Let $x$, $y$ and $z$ be random vectors, then denote the covariance matrix of $x$ and $y$ by $\Sigma_{xy}$ and the conditional covariance matrix of $x$ and $y$ conditioned on $z$ by $\Sigma_{xy|z}$. Let $a$ be a vector in a Hilbert space, and $\mathscr{B}$, $\mathscr{C}$ Hilbert spaces. Then, denote the orthogonal projection of $a$ onto $\mathscr{B}$ by $a/\mathscr{B}$ and the oblique projection of $a$ onto $\mathscr{B}$ along $\mathscr{C}$ by $a/_{\mathscr{C}}\mathscr{B}$. Let $A$ be an $[m \times n]$ matrix, then $\mathcal{L}\{A\} := \{A\boldsymbol{\alpha}|\boldsymbol{\alpha} \in \mathbb{R}^n\}$ will be referred to as the column space and $\mathcal{L}\{A'\} := \{A'\boldsymbol{\alpha}|\boldsymbol{\alpha} \in \mathbb{R}^m\}$ the row space of $A$. $\bullet'$ denotes the transpose of a matrix $\bullet$, and $I_d \in \mathbb{R}^{d \times d}$ is the identity matrix.

## 2  Rationales

### 2.1  Problem Description and Some Definitions

Consider two discrete-time wide-sense stationary vector processes $\{\boldsymbol{u}(t), \boldsymbol{y}(t), t = 0, \pm 1, \cdots\}$ with dimensions $n_u$ and $n_y$, respectively. The first component $\boldsymbol{u}(t)$ models the *input* signal while the second component $\boldsymbol{y}(t)$ models the *output* of the unknown stochastic system, which we want to construct from observed input-output data, as a nonlinear state-space system:

$$\begin{aligned}
\boldsymbol{x}(t+1) &= \boldsymbol{g}(\boldsymbol{x}(t), \boldsymbol{u}(t)) + \boldsymbol{v} \\
\boldsymbol{y}(t) &= \boldsymbol{h}(\boldsymbol{x}(t), \boldsymbol{u}(t)) + \boldsymbol{w},
\end{aligned} \tag{1}$$

where $\boldsymbol{x}(t) \in \mathbb{R}^n$ is the state vector, and $\boldsymbol{v}$ and $\boldsymbol{w}$ are the system and observation noises. Throughout this paper, we shall assume that the joint process $(\boldsymbol{u}, \boldsymbol{y})$ is a stationary and purely nondeterministic full rank process [3,5,10]. It is also assumed that the two processes are zero-mean and have finite joint covariance matrices. A basic step in solving this realization problem, which is also the core of the subspace identification algorithm presented later, is the construction of a *state space* of the system. In this paper, we will derive a practical algorithm for this problem based on stochastic realization with reproducing kernel Hilbert spaces.

We denote the joint input-output process $\boldsymbol{w}(t)' = [\boldsymbol{y}(t)', \boldsymbol{u}(t)'] \in \mathbb{R}^{n_w} (n_w = n_u + n_y)$ and feature maps $\boldsymbol{\phi}_u : \mathscr{U}_t \to \mathcal{F}_u \in \mathbb{R}^{n_{\phi u}}$, $\boldsymbol{\phi}_y : \mathscr{Y}_t \to \mathcal{F}_y \in \mathbb{R}^{n_{\phi y}}$ and $\boldsymbol{\phi}_w : \mathscr{W}_t \to \mathcal{F}_w \in \mathbb{R}^{n_{\phi w}}$ with the Mercer kernels $k_u$, $k_y$ and $k_w$, where $\mathscr{U}_t$, $\mathscr{Y}_t$ and $\mathscr{W}_t$ are the Hilbert spaces generated by the second-order random variables $\boldsymbol{u}(t)$, $\boldsymbol{y}(t)$ and $\boldsymbol{w}(t)$, and $\mathcal{F}_y$, $\mathcal{F}_u$ and $\mathcal{F}_w$ are the respective feature spaces. Moreover, we define the future output, input and the past input-output vectors in the feature spaces as

$$\begin{aligned}
\boldsymbol{f}^\phi(t) &:= \left[\boldsymbol{\phi}_y(\boldsymbol{y}(t))', \boldsymbol{\phi}_y(\boldsymbol{y}(t+1))', \cdots, \boldsymbol{\phi}_y(\boldsymbol{y}(t+l-1))'\right]' \in \mathbb{R}^{ln_y^\phi}, \\
\boldsymbol{u}_+^\phi(t) &:= \left[\boldsymbol{\phi}_u(\boldsymbol{u}(t))', \boldsymbol{\phi}_u(\boldsymbol{u}(t+1))', \cdots, \boldsymbol{\phi}_u(\boldsymbol{u}(t+l-1))'\right]' \in \mathbb{R}^{ln_u^\phi}, \\
\boldsymbol{p}^\phi(t) &:= \left[\boldsymbol{\phi}_w(\boldsymbol{w}(t-1))', \boldsymbol{\phi}_w(\boldsymbol{w}(t-2))', \cdots\right]' \in \mathbb{R}^\infty,
\end{aligned} \tag{2}$$

and the Hilbert spaces generated by these random variables as:

$$\mathscr{P}_t^\phi = \overline{\mathrm{span}}\{\boldsymbol{\phi}(\boldsymbol{w}(\tau))|\tau < t\}, \ \mathscr{U}_t^{\phi+} = \overline{\mathrm{span}}\{\boldsymbol{\phi}(\boldsymbol{u}(\tau))|\tau \geq t\}, \ \mathscr{Y}_t^{\phi+} = \overline{\mathrm{span}}\{\boldsymbol{\phi}(\boldsymbol{y}(\tau))|\tau \geq t\}. \tag{3}$$

$\mathscr{U}_t^{\phi-}$ and $\mathscr{Y}_t^{\phi-}$ are defined similarly. These spaces are assumed to be closed with respect to the root-mean-square norm $\|\xi\| := [E\{\xi^2\}]^{1/2}$, where $E\{\cdot\}$ denotes the expectation value, and thus are thought of as Hilbert subspaces of an ambient Hilbert space $\mathscr{H}^\phi := \mathscr{U}^\phi \vee \mathscr{Y}^\phi$ containing all linear functionals of the joint process in the feature spaces $(\boldsymbol{\phi}_u(\boldsymbol{u}), \boldsymbol{\phi}_y(\boldsymbol{y}))$.

### 2.2  Optimal Predictor in Kernel Feature Space

First, we require the following technical assumptions [3,5].

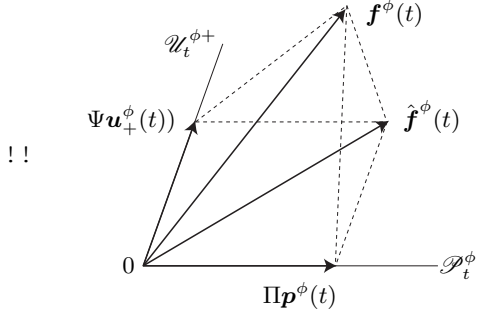

!!

Figure 1: Optimal predictor $\hat{\boldsymbol{f}}^{\phi}(t)$ of future output in feature space based on $\mathscr{P}_t^{\phi} \vee \mathscr{U}_t^{+\phi}$.

ASSUMPTION 1. *The input $\boldsymbol{u}$ is 'exogenous', i.e., no feedback from the output $\boldsymbol{y}$ to the input $\boldsymbol{u}$.*

ASSUMPTION 2. *The input process is 'sufficiently rich'. More precisely, at each time $t$, the input space $\mathscr{U}_t$ has the direct sum decomposition $\mathscr{U}_t = \mathscr{U}_t^- + \mathscr{U}_t^+$ $\left(\mathscr{U}_t^- \cap \mathscr{U}_t^+ = \{0\}\right)$.*

Note that assumption 2 implies that the input process is purely nondeterministic and admits a spectral density matrix without zeros on the unit circle (i.e., coercive). This is too restrictive in many practical situations and we can instead assume only a persistently exciting (PE) condition of sufficiently high order and finite dimensionality for an underlying "true" system from the outset. Then, we can give the following proposition which enables us to develop a subspace method in feature space, as in the linear case.

PROPOSITION 1. *If assumptions 1 and 2 are satisfied, then similar conditions in the feature spaces described below are fulfilled:*

*(1) There is no feedback from $\phi_y(\boldsymbol{y})$ to $\phi_u(\boldsymbol{u})$.*

*(2) $\mathscr{U}_t^{\phi}$ has the direct sum decomposition $\mathscr{U}_t^{\phi} = \mathscr{U}_t^{\phi-} + \mathscr{U}_t^{\phi+}$ $(\mathscr{U}_t^{\phi-} \cap \mathscr{U}_t^{\phi+} = \{0\})$*

PROOF. Condition (2) is shown straightforwardly from assumption 2 and the properties of the reproducing kernel Hilbert spaces. As $\mathscr{U}_t^+ \perp \mathscr{Y}_t^- | \mathscr{U}_t^-$ (derived from assumption 1) and $\mathscr{Y}^-/\mathscr{U}_t^+ \vee \mathscr{U}_t^- = \mathscr{Y}_t^-/\mathscr{U}_t^-$ are equivalent, if the orthogonal complement of $\mathscr{U}_t$ is denoted by $\mathscr{U}_t^{\perp}$, we can obtain $\mathscr{Y}_t^- = \mathscr{U}_t^- + \mathscr{U}_t^{\perp}$. Now, when representing $\mathscr{Y}_t^{-\phi}$ using the input space on feature space $\mathscr{U}_t^{\phi}$ and the orthogonal complement $\mathscr{U}_t^{\perp\phi}$, we can write $\mathscr{Y}_t^{-\phi} = \mathscr{U}_t^{-\phi} + \mathscr{U}_t^{\perp\phi}$ because $\mathscr{U}_t^{\phi} = \mathscr{U}_t^{-\phi} + \mathscr{U}_t^{+\phi}$ from condition (2), $\mathscr{U}_t^+ \perp \mathscr{U}_t^{\perp}$, and owing to the properties of the reproducing kernel Hilbert spaces. Therefore, $\mathscr{U}_t^{+\phi} \perp \mathscr{Y}_t^{-\phi} | \mathscr{U}_t^{-\phi}$ can be shown by tracing inversely. $\square$

Using proposition 1, we now obtain the following representation result.

THEOREM 1. *Under assumptions 1 and 2, the optimal predictor $\hat{\boldsymbol{f}}^{\phi}(t)$ of the future output vector in feature space $\boldsymbol{f}^{\phi}(t)$ based on $\mathscr{P}_t^{\phi} \vee \mathcal{U}_t^{\phi+}$ is uniquely given by the sum of the oblique projections:*

$$\hat{\boldsymbol{f}}^{\phi}(t) = \boldsymbol{f}^{\phi}(t) / \mathscr{P}_t^{\phi} \vee \mathcal{U}_t^{\phi+} = \Pi \boldsymbol{p}^{\phi}(t) + \Psi \boldsymbol{u}_+^{\phi}(t), \tag{4}$$

*in which $\Pi$ and $\Psi$ satisfy the discrete Wiener-Hopf-type equations*

$$\Pi \Sigma_{\phi_p \phi_p | \phi_u} = \Sigma_{\phi_f \phi_p | \phi_u}, \quad \Psi \Sigma_{\phi_u \phi_u | \phi_p} = \Sigma_{\phi_f \phi_u | \phi_p}. \tag{5}$$

PROOF. From proposition 1, the proof can be carried out as in the linear case (cf. [3,5]). $\square$

## 2.3 Construction of State Vector

Let $L_f$, $L_p$ be the square root matrices of $\Sigma_{\phi_f \phi_f | \phi_u}$, $\Sigma_{\phi_p \phi_p | \phi_u}$, i.e., $\Sigma_{\phi_f \phi_f | \phi_u} = L_f L_f'$, $\Sigma_{\phi_p \phi_p | \phi_u} = L_p L_p'$, and assume that the SVD of the normalized conditional covariance is given by

$$L_f^{-1} \Sigma_{\phi_f \phi_p | \phi_u} (L_p^{-1})' = U S V', \tag{6}$$

where $S \in \mathbb{R}^{ln_{\phi_y} \times n_{\phi_p}}$ is the matrix with all entries being zero, except the leading diagonal, which has the entries $\rho_i$ satisfying $\rho_1 \geq \cdots \geq \rho_n > 0$ for $n = \min(ln_{\phi_y}, n_{\phi_p})$, and $U$, $V$ are square orthogonal.

We define the extended observability and controllability matrices

$$\mathcal{O} := L_f U S^{1/2}, \quad \mathcal{C} := S^{1/2} V' L_p', \tag{7}$$

where $\mathrm{rank}(\mathcal{O}) = \mathrm{rank}(\mathcal{R}) = n$. Then, from the SVD of Eq. (6), the block Hankel matrix $\Sigma_{\phi_f \phi_p | \phi_u}$ has the classical rank factorization $\Sigma_{\phi_f \phi_p | \phi_u} = \mathcal{OC}$. If a 'state vector' is now defined to be the $n$-dimensional vector

$$\boldsymbol{x}(t) = \mathcal{C} \Sigma_{\phi_p \phi_p | \phi_u}^{-1} \boldsymbol{p}^\phi(t) = S^{1/2} V' L_p^{-1} \boldsymbol{p}^\phi(t), \tag{8}$$

it is readily seen that $\boldsymbol{x}(t)$ is a basis for the stationary oblique predictor space $\mathscr{X}_t := \mathscr{Y}_t^{\phi+} /_{\mathscr{U}_t^{\phi+}} \mathscr{P}_t^\phi$, which, on the basis of general geometric principles, can be shown to be a minimal state space for the process $\boldsymbol{\phi}_y(\boldsymbol{y})$, as in the linear case [3,5]. This is also assured by the fact that the oblique projection of $\boldsymbol{f}^\phi(t)$ onto $\mathscr{U}_t^{\phi+}$ along $\mathscr{P}_t^\phi$ can be expressed, using Eqs. (5), (7) and (8), as

$$\boldsymbol{f}^\phi(t) /_{\mathscr{U}_t^{\phi+}} \mathscr{P}_t^\phi = \Pi \boldsymbol{p}^\phi(t) = \Sigma_{\phi_f \phi_p | \phi_u} \Sigma_{\phi_p \phi_p | \phi_u}^{-1} \boldsymbol{p}^\phi(t) = \mathcal{O}\boldsymbol{x}(t) \tag{9}$$

and $\mathrm{rank}(\mathcal{O}) = n$, and the variance matrix of $\boldsymbol{x}(t)$ is nonsingular. In terms of $\boldsymbol{x}(t)$, the optimal predictor $\hat{\boldsymbol{f}}^\phi(t)$ in Eq. (4) has the form

$$\hat{\boldsymbol{f}}^\phi = \mathcal{O}\boldsymbol{x}(t) + \Psi \boldsymbol{u}_+^\phi(t). \tag{10}$$

It is seen that $\boldsymbol{x}(t)$ is a conditional minimal sufficient statistic carrying exactly all the information contained in $\mathscr{P}_t^\phi$ that is necessary for estimating the future outputs, given the future inputs.

In analogy with the linear case [3,5], the output process in feature space $\boldsymbol{\phi}_y(\boldsymbol{y}(t))$ now admits a minimal stochastic realization with the state vector $\boldsymbol{x}(t)$ of the form

$$\begin{aligned} \boldsymbol{x}(t+1) &= A^\phi \boldsymbol{x}(t) + B^\phi \boldsymbol{\phi}_u(\boldsymbol{u}(t)) + K^\phi \boldsymbol{e}(t), \\ \boldsymbol{\phi}_y(\boldsymbol{y}(t)) &= C^\phi \boldsymbol{x}(t) + D^\phi \boldsymbol{\phi}_u(\boldsymbol{u}(t)) + \boldsymbol{e}(t), \end{aligned} \tag{11}$$

where $A^\phi \in \mathbb{R}^{n \times n}$, $B^\phi \in \mathbb{R}^{n \times n_{\phi_u}}$, $C^\phi \in \mathbb{R}^{n_{\phi_y} \times n}$, $D^\phi \in \mathbb{R}^{n_{\phi_y} \times n_{\phi_u}}$ and $K^\phi \in \mathbb{R}^{n \times n_{\phi_y}}$ are constant matrices and $\boldsymbol{e}(t) := \boldsymbol{\phi}(\boldsymbol{y}(t)) - (\boldsymbol{\phi}(\boldsymbol{y}(t)) | \mathscr{P}_t^\phi \vee \mathscr{U}_t^\phi)$ is the prediction error.

## 2.4 Preimage

In this section, we describe the state-space model for the output $\boldsymbol{y}(t)$ while the state-space model (11), derived in the previous section, represents the output in feature space $\boldsymbol{\phi}_y(\boldsymbol{y}(t))$. At first, we define the feature maps $\boldsymbol{\phi}_x : \mathscr{X}_t \mapsto \mathcal{F}_x \in \mathbb{R}^{n_{\phi_x}}$, $\boldsymbol{\phi}_{\bar{u}} := \mathscr{U}_t \mapsto \mathcal{F}_{\bar{u}} \in \mathbb{R}^{n_{\phi_{\bar{u}}}}$ and the linear space $\mathscr{X}_t^\phi$, $\bar{\mathscr{U}}_t^\phi$ generated by $\boldsymbol{\phi}_x(\boldsymbol{x}(t))$, $\boldsymbol{\phi}_{\bar{u}}(\boldsymbol{u}(t))$. Then, the product of $\mathscr{X}_t^\phi$ and $\bar{\mathscr{U}}_t^\phi$ satisfies $\mathscr{X}_t^\phi \cap \bar{\mathscr{U}}_t^\phi = 0$ because $\mathscr{X}_t \cap \mathscr{U}_t^\phi = 0$ and $\boldsymbol{\phi}_x$, $\boldsymbol{\phi}_{\bar{u}}$ are bijective. Therefore, the output $\boldsymbol{y}(t)$ is represented as the direct sum of the oblique projections as

$$\boldsymbol{y}(t) / \mathscr{X}_t^\phi \vee \bar{\mathscr{U}}_t^\phi = \bar{C}^\phi \boldsymbol{\phi}_x(\boldsymbol{x}(t)) + \bar{D}^\phi \boldsymbol{\phi}_{\bar{u}}(\boldsymbol{u}(t)). \tag{12}$$

As a result, we can obtain the following theorem.

THEOREM 2. Under assumptions 1 and 2, if $\mathrm{rank}\, \Sigma_{\phi_f \phi_p | \phi_u} = n$, then the output $\boldsymbol{y}$ can be represented in the following state-space model:

$$\begin{aligned} \boldsymbol{x}(t+1) &= A^\phi \boldsymbol{x}(t) + B^\phi \boldsymbol{\phi}_u(\boldsymbol{u}(t)) + \bar{K}^\phi \boldsymbol{\phi}_e(\bar{\boldsymbol{e}}(t)), \\ \boldsymbol{y}(t) &= \bar{C}^\phi \boldsymbol{\phi}_x(\boldsymbol{x}(t)) + \bar{D}^\phi \boldsymbol{\phi}_{\bar{u}}(\boldsymbol{u}(t)) + \bar{\boldsymbol{e}}(t), \end{aligned} \tag{13}$$

where $\bar{\boldsymbol{e}}(t) := \boldsymbol{y}(t) - \boldsymbol{y}(t) / \mathscr{X}_t^\phi \vee \bar{\mathscr{U}}_t^\phi$ is the prediction error and $\bar{K}^\phi := K^\phi A_{\bar{e}}$, in which $A_{\bar{e}}$ is the coefficient matrix of the nonlinear regression from $\bar{\boldsymbol{e}}(t)$ to $\boldsymbol{e}(t)$ [1].

# 3 Approximations

## 3.1 Realization with Finite Data

In practice, the state vector and associated state-space model should be constructed with available finite data. Let the past vector $\boldsymbol{p}^\phi(t)$ be truncated to finite length, i.e., $\boldsymbol{p}_T^\phi(t) := [\boldsymbol{\phi}_w(\boldsymbol{w}(t-1))', \boldsymbol{\phi}_w(\boldsymbol{w}(t-2))', \cdots, \boldsymbol{\phi}_w(\boldsymbol{w}(t-T))']' \in \mathbb{R}^{T(n_{\phi_y}+n_{\phi_u})}$, where $T > 0$, and define $\mathscr{P}_{[t-T,t)} :=$ $\text{span}\{\boldsymbol{p}_T^\phi(\tau)|\tau < t\}$. Then, the following theorem describes the construction of the state vector and the corresponding state-space system which form the finite-memory predictor $\hat{\boldsymbol{f}}_T^\phi(t) :=$ $\boldsymbol{f}^\phi(t)/\mathscr{U}_t^{\phi+} \cap \mathscr{P}_{[t-T,t)}^\phi$.

THEOREM 3. Under assumptions 1 and 2, if $\text{rank}(\Sigma_{\phi_f\phi_p|\phi_u}) = n$, then the process $\boldsymbol{\phi}_y(\boldsymbol{y})$ is expressed by the following nonstationary state-space model:

$$\begin{aligned}
\hat{\boldsymbol{x}}_T(t+1) &= A^\phi\hat{\boldsymbol{x}}_T(t) + B^\phi\boldsymbol{\phi}_u(\boldsymbol{u}(t)) + K^\phi(t)\hat{\boldsymbol{e}}_T(t), \\
\boldsymbol{\phi}_y(\boldsymbol{y}(t)) &= C^\phi\hat{\boldsymbol{x}}_T(t) + D^\phi\boldsymbol{\phi}_u(\boldsymbol{u}(t)) + \hat{\boldsymbol{e}}_T(t).
\end{aligned} \tag{14}$$

where the state vector $\hat{\boldsymbol{x}}_T(t)$ is a basis on the finite-memory predictor space $\mathscr{U}_t^{\phi+}/_{\mathscr{U}_t^{\phi+}}\mathscr{P}_{[t-T,t)}^\phi$, and $\hat{\boldsymbol{e}}_T(t) := \boldsymbol{\phi}_y(\boldsymbol{y}(t)) - (\boldsymbol{\phi}_y(\boldsymbol{y}(t))|\mathscr{P}_{[T,t)}^\phi \vee \mathscr{U}_t^{\phi+})$ is the prediction error.

The proof can be carried out as in the linear case (cf. [3,5]). In other words, we can obtain the approximated state vector $\hat{\boldsymbol{x}}_T$ by applying the facts in Section 2 to finite data. This state vector differs from $\boldsymbol{x}(t)$ in Eq. (8); however, when $T \to \infty$, the difference between $\hat{\boldsymbol{x}}_T(t)$ and $\boldsymbol{x}(t)$ converges to zero and the covariance matrix of the estimation error $P^\phi$ also converges to the stabilizing solution of the following Algebra Riccati Equation (ARE):

$$P^\phi = A^\phi P^\phi A^{\phi'} + \Sigma_w^\phi\Sigma_w^{\phi'} - (A^\phi P^\phi C^{\phi'} + \Sigma_w^\phi\Sigma_w^{\phi'})(C^\phi P^\phi C^{\phi'} + \Sigma_e^\phi\Sigma_e^{\phi'})^{-1}(A^\phi P^\phi C^{\phi'} + \Sigma_w^\phi\Sigma_w^{\phi'})'. \tag{15}$$

Moreover, the Kalman gain $K^\phi$ converges to

$$K^\phi = (A^\phi P^\phi C^{\phi'} + \Sigma_w^\phi\Sigma_w^{\phi'})(C^\phi P^\phi C^{\phi'} + \Sigma_e^\phi\Sigma_e^{\phi'})^{-1}, \tag{16}$$

where $\Sigma_w^\phi$ and $\Sigma_e^\phi$ are the covariance matrices of errors in the state and observation equations, respectively.

## 3.2 Using Kernel Principal Components

Let $\boldsymbol{z}$ be a random variable, $k_z$ a Mercer kernel with a feature map $\boldsymbol{\phi}_z$ and a feature space $\mathcal{F}_z$, and denote $\Phi_z := [\boldsymbol{\phi}_z(\boldsymbol{z}_1), \cdots, \boldsymbol{\phi}_z(\boldsymbol{z}_m)]'$ and the associated Gram matrix $G_z := \Phi_z\Phi_z'$. The first $i$th principal components $\boldsymbol{u}_{z,i} \in \mathcal{L}\{\Phi_z'\}(i = 1, \cdots, d_z)$ combined in a matrix $U_z = [\boldsymbol{u}_{z,1}, \cdots, \boldsymbol{u}_{z,d_z}]$ form an orthonormal basis of a $d_z$-dimensional subspace $\mathcal{L}\{U_z\} \subseteq \mathcal{L}\{\Phi_z'\}$, and can therefore also be described as the linear combination $U_z = \Phi_z'A_z$, where the matrix $A_z \in \mathbb{R}^{m \times d_z}$ holds the expansion coefficients. $A_z$ is found by, for example, the eigendecomposition $G_z = \Gamma_z\Lambda_z\Gamma_{-z}'$ such that $A_z$ consists of the first $d_z$ columns of $\Gamma_z\Lambda_z^{-1/2}$. Then, $\Phi_z$ with respect to the principal components is given by $C_z := \Phi_zU_z = \Phi_z\Phi_z'A_z = G_zA_z$ [11]. From the orthogonality of $\Gamma_z$ (i.e., $\Gamma_z'\Gamma_z = \Gamma_z\Gamma_z' = I_m$), we can derive the following equation:

$$(A_z'G_zG_zA_z)^{-1} = \left((\Gamma_z\Lambda_{z,d}^{-1/2})'(\Gamma_z\Lambda_z\Gamma_z')(\Gamma_z\Lambda_z\Gamma_z')(\Gamma_z\Lambda_{z,d}^{-1/2})\right)^{-1} = \bar{A}_z'G_z^{-1}G_z^{-1}\bar{A}_z, \tag{17}$$

where $\Lambda_{z,d}$ is the matrix which consists of the first $d_z$ columns of $\Lambda_z$, and $\bar{A}_z := \Gamma_z\Lambda_{z,d}^{1/2}$ satisfying $\bar{A}_z'A_z = A_z'\bar{A}_z = I_{d_z}$ and $\bar{A}_zA_z' = A_z\bar{A}_z' = I_m$.

This property of kernel principal components enables us to approximate matters described in the previous sections in computable forms. First, using Eq. (17), the conditional covariance matrix $\Sigma_{\phi_f\phi_f|\phi_u}$ can be expressed as

$$\begin{aligned}
\Sigma_{\phi_f\phi_f|\phi_u} &= \Sigma_{\phi_f\phi_f} - \Sigma_{\phi_f\phi_u}\Sigma_{\phi_u\phi_u}^{-1}\Sigma_{\phi_u\phi_f} \\
&\approx A_f'G_fG_fA_f - (A_f'G_fG_uA_u)(A_u'G_uG_uA_u)^{-1}(A_u'G_uG_fA_f) \\
&= A_f'\left(G_fG_f - G_fG_u(G_uG_u)^{-1}G_uG_f\right)A_f(:= A_f'\hat{\Sigma}_{ff|u}A_f),
\end{aligned} \tag{18}$$

where $\hat{\Sigma}_{ff|u}$ may be called the empirical conditional covariance operators, and the regularized variant can be obtained by replacing $G_f G_f$, $G_u G_u$ with $(G_f + \epsilon I_m)^2$, $(G_u + \epsilon I_m)^2$ ($\epsilon > 0$) (cf.[12,13]). $\Sigma_{\phi_p \phi_p | \phi_u}$ and $\Sigma_{\phi_f \phi_p | \phi_u}$ can be approximated as well. Moreover, using $L_*^{-1} = \hat{L}_*^{-1} \bar{A}_*$, where $\hat{L}_*$ is the square root matrix of $\hat{\Sigma}_{\phi_* \phi_* | u}$ ($* = p, f$) [2], we can represent Eqs. (6) and (8) approximately as

$$L_f^{-1} \Sigma_{\phi_f \phi_p | \phi_u} (L_p^{-1})' \approx (\hat{L}_f^{-1} \bar{A}_f)(A_f' \hat{\Sigma}_{fp|u} A_p)\left(\bar{A}_p'(\hat{L}_p^{-1})'\right) = \hat{L}_f^{-1} \hat{\Sigma}_{fp|u}(\hat{L}_p^{-1})' = \hat{U} \hat{S} \hat{V}', \quad (19)$$

$$\boldsymbol{x}(t) = S^{1/2} V' L_p^{-1} \boldsymbol{p}^\phi(t) \approx \hat{S}^{1/2} \hat{V}'(\hat{L}_p^{-1} \bar{A}_p)(A_p' \boldsymbol{k}(\boldsymbol{p}(t))) = \hat{S}^{1/2} \hat{V}' \hat{L}_p^{-1} \boldsymbol{k}(\boldsymbol{p}(t)), \quad (20)$$

where $\boldsymbol{k}(\boldsymbol{p}(t)) := \Phi_p \boldsymbol{p}^\phi(t) = [k_p(\boldsymbol{p}_1(t), \boldsymbol{p}(t)), \cdots, k_p(\boldsymbol{p}_m(t), \boldsymbol{p}(t))]'$.

In addition, we can apply this approximation with the kernel PCA to the state-space models derived in the previous sections. First, Eq. (11) can be approximated as

$$\boldsymbol{x}(t+1) = A^\phi \boldsymbol{x}(t) + B^\phi A_u' \boldsymbol{k}_u(\boldsymbol{u}(t)) + K^\phi \boldsymbol{e}(t),$$
$$A_y' \boldsymbol{k}_y(\boldsymbol{y}(t)) = C^\phi \boldsymbol{x}(t) + D^\phi A_u' \boldsymbol{k}_u(\boldsymbol{u}(t)) + \boldsymbol{e}(t), \quad (21)$$

where $A_u$ and $A_y$ are the expansion coefficient matrices found by the eigendecomposition of $G_u$ and $G_y$, respectively. Also, using the coefficient matrices $A_x$, $A_e$ and $A_{\bar{u}}$, Eq.(13) can be written as

$$\boldsymbol{x}(t+1) = A^\phi \boldsymbol{x}(t) + B^\phi A_u' \boldsymbol{k}_u(\boldsymbol{u}(t)) + \bar{K}^\phi A_e' \boldsymbol{k}_e(\bar{\boldsymbol{e}}(t)),$$
$$\boldsymbol{y}(t) = \bar{C}^\phi A_x' \boldsymbol{k}_x(\boldsymbol{x}(t)) + \bar{D}^\phi A_{\bar{u}}' \boldsymbol{k}_u(\boldsymbol{u}(t)) + \bar{\boldsymbol{e}}(t). \quad (22)$$

## 4 Algorithm

In this section, we give a subspace identification algorithm based on the discussions in the previous sections. Denote the finite input-output data as $\{\boldsymbol{u}(t), \boldsymbol{y}(t), t = 1, 2, \cdots, N + 2l - 1\}$, where $l > 0$ is an integer larger than the dimension of system $n$ and $N$ is the *sufficient* large integer, and assume that all data is centered. First, using the Gram matrices $G_u$, $G_y$ and $G_w$ associated with the input, the output, and the input-output, repectively, we must to calculate the Gram matrices $G_U$, $G_Y$ and $G_W$ corresponding to the past input, the future output, and the past input-output defined as

$$G_U := \begin{bmatrix} \sum_{i=l+1}^{2l} G_{u,ii} & \sum_{i=l+1}^{2l} G_{u,i(i+1)} & \cdots & \sum_{i=l+1}^{2l} G_{u,i(i+N-1)} \\ \sum_{i=l+1}^{2l} G_{u,(i+1)i} & \sum_{i=l+1}^{2l} G_{u,(i+1)(i+1)} & \cdots & \sum_{i=l+1}^{2l} G_{u,(i+1)(i+N-1)} \\ \vdots & \vdots & \ddots & \vdots \\ \sum_{i=l+1}^{2l} G_{u,(i+N-1)i} & \sum_{i=l+1}^{2l} G_{u,(i+N-1)(i+1)} & \cdots & \sum_{i=l+1}^{2l} G_{u,(i+N-1)(i+N-1)} \end{bmatrix}, \quad (23)$$

$$G_W := \begin{bmatrix} \sum_{i=1}^{l} G_{w,ii} & \sum_{i=1}^{l} G_{w,i(i+1)} & \cdots & \sum_{i=1}^{l} G_{w,i(i+N-1)} \\ \sum_{i=1}^{l} G_{w,(i+1)i} & \sum_{i=1}^{l} G_{w,(i+1)(i+1)} & \cdots & \sum_{i=1}^{l} G_{w,(i+1)(i+N-1)} \\ \vdots & \vdots & \ddots & \vdots \\ \sum_{i=1}^{l} G_{w,(i+N-1)i} & \sum_{i=1}^{l} G_{w,(i+N-1)(i+1)} & \cdots & \sum_{i=1}^{l} G_{w,(i+N-1)(i+N-1)} \end{bmatrix}, \quad (24)$$

and $G_Y$ is defined analogously to $G_U$. Now the procedure is given as follows.

**Step 1** Calculate the regularized empirical covariance operators and their square root matrices as

$$\hat{\Sigma}_{ff|u} = (G_Y + \epsilon I_N)^2 - G_Y G_U (G_U + \epsilon I_N)^{-2} G_U G_Y = \hat{L}_f \hat{L}_f',$$
$$\hat{\Sigma}_{pp|u} = (G_W + \epsilon I_N)^2 - G_W G_U (G_U + \epsilon I_N)^{-2} G_U G_W = \hat{L}_p \hat{L}_p', \quad (25)$$
$$\hat{\Sigma}_{fp|u} = G_Y G_W - G_Y G_U (G_U + \epsilon I_N)^{-2} G_U G_W.$$

**Step 2** Calculate the SVD of the normalized covariance matrix (cf. Eq. (19))

$$L_f^{-1}\hat{\Sigma}_{fp|u}(\hat{L}_p^{-1})' = \hat{U}\hat{S}\hat{V}' \approx U_1 S_1 V_1, \tag{26}$$

where $S_1$ is obtained by neglecting the small singular values so that the dimension of the state vector $n$ equals the dimension of $S_1$.

**Step 3** Estimate the state sequence as (cf. Eq. (20))

$$X_l := [\boldsymbol{x}(l), \boldsymbol{x}(l+1), \cdots, \boldsymbol{x}(l+N-1)] = S_1^{1/2}V_1'\hat{L}_p^{-1}G_W, \tag{27}$$

and define the following matrices consisting of $N-1$ columns:

$$\hat{X}_{l+1} = \bar{X}_l(:,2:N), \quad \hat{X}_l = \bar{X}_l(:,1:N-1). \tag{28}$$

**Step 4** Calculate the eigendecomposition of the Gram matrices $G_u$, $G_{\bar{u}}$, $G_y$ and $G_x$ and the corresponding expansion coefficient matrices $A_u$, $A_{\bar{u}}$, $A_y$ and $A_x$. Then, determine the system matrices $A^\phi$, $B^\phi$, $C^\phi$, $D^\phi$, $\bar{C}^\phi$ and $\bar{D}^\phi$ by applying regularized least square regressions to the following equations (cf. Eqs. (21) and (22)):

$$\begin{bmatrix} \hat{X}_{k+1} \\ A_y'G_y(:,2,N) \end{bmatrix} = \begin{bmatrix} A^\phi & B^\phi \\ C^\phi & D^\phi \end{bmatrix} \begin{bmatrix} \hat{X}_k \\ A_u'G_u(:,1,N-1) \end{bmatrix} + \begin{bmatrix} \rho_w \\ \rho_e \end{bmatrix}, \tag{29}$$

$$Y_{l|l} = \bar{C}^\phi(A_x'G_x(:,2,N)) + \bar{D}^\phi(\bar{A}_u'G_u(:,2,N)) + \bar{\rho}_e, \tag{30}$$

where the matrices $\rho_w$, $\rho_e$ and $\bar{\rho}_e$ are the residuals.

**Step 5** Calculate the covariance matrices of the residuals

$$\begin{bmatrix} \Sigma_w & \Sigma_{we} \\ \Sigma_{ew} & \Sigma_e \end{bmatrix} = \frac{1}{N-1} \begin{bmatrix} \rho_w\rho_w' & \rho_w\rho_e' \\ \rho_e\rho_w' & \rho_e\rho_e' \end{bmatrix}, \tag{31}$$

solve ARE (15), and, using the stabilizing solution, calculate the Kalman gain $K^\Phi$ in Eq. (16).

## 5 Simulation Result

In this section, we illustrate the proposed algorithm for learning nonlinear dynamical systems with synthetic data. The data was generated by simulating the following system [8] using the 4th- and 5th-order Runge-Kutta method with a sampling time of 0.05 seconds:

$$\begin{aligned} \dot{x}_1(t) &= x_2(t) - 0.1\cos(x_1(t))(5x_1(t) - 4x_1^3(t) + x_1^5(t)) - 0.5\cos(x_1(t))u(t), \\ \dot{x}_2(t) &= -65x_1(t) + 50x_1^3(t) - 15x_1^5(t) - x_2(t) - 100u(t), \\ y(t) &= x_1(t), \end{aligned} \tag{32}$$

where the input was a zero-order-hold white noise signal uniformly distributed between $-0.5$ and $0.5$. We applied our algorithm on a set of 600 data points, and then validated the obtained model using a fresh data set of 400 points. As a kernel function, we used the RBF Gaussian kernel $k(\boldsymbol{z}_i, \boldsymbol{z}_j) = \exp(-\|\boldsymbol{z}_i - \boldsymbol{z}_j\|^2/2\sigma_z)$. The parameters to be tuned for our method are thus the widths of the kernels $\sigma$ for $\boldsymbol{u}$, $\boldsymbol{y}$, $\boldsymbol{w}$ and $\boldsymbol{x}$, the regularization degree $\epsilon$, and the row-block number $l$ of the Hankel matrix. In addition, we must select the order of the system and the number of kernel principal components $n_*^{pc}$ for $\boldsymbol{u}$, $\boldsymbol{y}$ and $\boldsymbol{e}$. Figure 2 shows free-run simulation results of the model acquired by our algorithm, in which the parameters were set as $\sigma_u = 2.5$, $\sigma_y = 3.5$, $\sigma_w = 4.5$, $\sigma_x = 1.0$, $n_u^{pc} = n_y^{pc} = 4$, $n_x^{pc} = 9$ and $\epsilon = 0.05$, and, for comparison, by the linear subspace identification [5]. The row-block number $l$ was set as 10 in both identifications. The simulation errors [2]

$$\epsilon = \frac{100}{n_y} \sum_{c=1}^{n_y} \sqrt{\frac{\sum_{i=1}^m((y_i)_c - (y_i^s)_c)^2}{\sum_{j=1}^m((y_i)_c)^2}}, \tag{33}$$

where $\boldsymbol{y}_i^s$ are simulated values and the used initial state is a least square estimation with the initial few points, were improved to 40.2 for our algorithm, from 44.1 for the linear method. The accuracy was improved by about 10 percent. The system orders are 8 for our algorithm, whle 10 for the linear method, in this case. We can see that our method can estimate the state sequence with more information and yield the model capturing the dynamics more precisely. However, the parameters involved much time and effort for tuning.

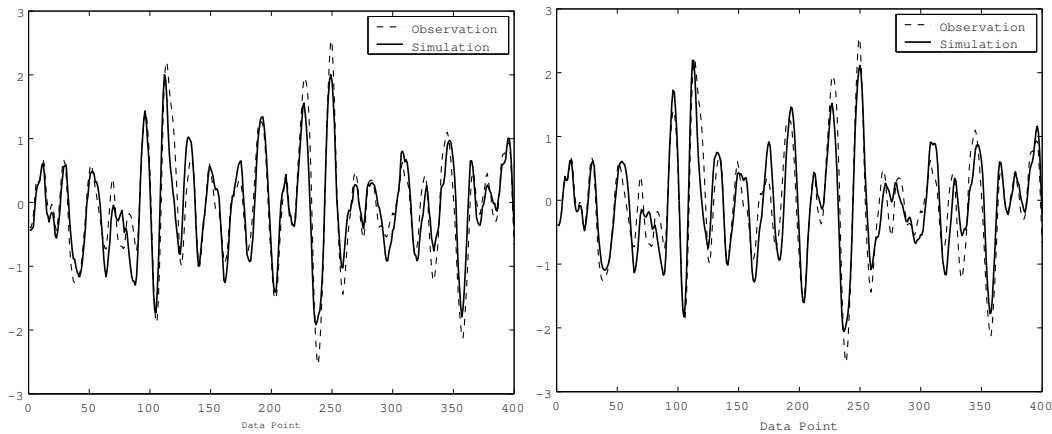

Figure 2: Comparison of simulated outputs. Left: Kernel subspace identification method (proposed method). Right: Linear subspace identification method [5]. The broken lines represent the observations and the solid lines represent the simulated values.

## 6  Conclusion

A new subspace method for learning nonlinear dynamical systems using reproducing kernel Hilbert spaces has been proposed. This approach is based on approximated solutions of two discrete Wiener-Hopf equations by covariance factorization in kernel feature spaces. The algorithm needs no iterative optimization procedures, and hence, solutions can be obtained in a fast and reliable manner. The comparative empirical results showed the high performance of our method. However, the parameters involved much time and effort for tuning. In future work, we will develop the idea for closed-loop systems for the identification of more realistic applications. Moreover, it should be possible to extend other established subspace identification methods to nonlinear frameworks as well.

**Acknowledgments**
The present research was supported in part through the 21st Century COE Program, "Mechanical Systems Innovation," by the Ministry of Education, Culture, Sports, Science and Technology.

**References**
[1] Roweis, S. & Ghahramani, Z. (1999) "A Unifying Review of Linear Gaussian Models" *Neural Computation*, **11** (2) : 305-345.

[2] Van Overschee, P. & De Moor, B. (1996) "Subspace Identification for Linear Systems: Theory, Implementation, Applications" *Kluwer Academic Publishers*, Dordrecht, Netherlands.

[3] Katayama, T. (2005) "Subspace Methods for System Identification: A Realization Approach" *Communications and Control Engineering, Springer Verlag*, 2005.

[4] Moonen, M. & Moor, B. D. & Vandenberghe, L. & Vandewalle, J. (1989) "On- and Off-line Identification of Linear State Space Models" *International Journal of Control*, **49** (1) : 219-232.

[5] Katayama, T. & Picci, G. (1999) "Realization of Stochastic Systems with Exogenous Inputs and Subspace Identification Methods" *Automatica*, **35** (10) : 1635-1652.

[6] Goethals, I. & Pelckmans, K. & Suykens, J. A. K. & Moor, B. D. (2005) "Subspace Identification of Hammerstein Systems Using Least Squares Support Vector Machines" *IEEE Trans. on Automatic Control*, **50** (10) : 1509-1519.

[7] Ni, X. & Verhaegen, M. & Krijgsman, A. & Verbruggen, H. B. (1996) "A New Method for Identification and Control of Nonlinear Dynamic Systems" *Engineering Application of Artificial Intelligence*, **9** (3) : 231-243.

[8] Verdult, V. & Suykens, J. A. K. & Boets, J. & Goethals, I. & Moor, B. D. (2004) "Least Squares Support Vector Machines for Kernel CCA in Nonlinear State-Space Identification" *Proceedings of the 16th International Symposium on Mathematical Theory of Networks and Systems*, (MTNS2004).

[9] Schölkopf, B. & Smola, A. (2002) "Learning with Kernels" *MIT Press*.

[10] Rozanov, N. I. (1963) "Stationary Random Processes" *Holden-Day*, San Francisco, CA.

[11] Kuss, M. & Graepel, T. (2003) "The Geometry of Kernel Canonical Correlation Analysis" *Technical Report, Max Planck Institute for Biological Cybernetics, Tubingen, Germany* (108).

[12] Bach, F. R., & Jordan, M. I. (2002) "Kernel Independent Component Analysis" *Journal of Machine Learning Research (JMLR)*, **3** : 1-48.

[13] Fukumizu, K. & Bach, F. R., & Jordan, M. I. (2004) "Dimensionality Reduction for Supervised Learning with Reproducing Kernel Hilbert Spaces" *Journal of Machine Learning Research (JMLR)*, **5** : 73-99.

## Footnotes

[1]Let $f$ be a map from $\bar{\boldsymbol{e}}(t)$ to $\boldsymbol{e}$ and minimize a regularized risk $c((\bar{\boldsymbol{e}}_1, \boldsymbol{e}_1, f(\bar{\boldsymbol{e}}_1)), \cdots, (\bar{\boldsymbol{e}}_m, \boldsymbol{e}_m, f(\bar{\boldsymbol{e}}_m))) + \Omega(\|f\|_{\mathscr{H}})$, where $\Omega : [0, \infty) \to \mathbb{R}$ is a strictly monotonically increasing function and $c : (\bar{\mathscr{E}} \times \mathbb{R}^2)^m \to \mathbb{R} \cup \{\infty\}$ ($\bar{\mathscr{E}} \in \mathrm{span}\{\bar{\boldsymbol{e}}\}$) is an arbitrary loss function; then, from the representer theorem[9], $f$ satisfies $f \in \mathrm{span}\{\boldsymbol{\phi}_e(\bar{\boldsymbol{e}}(t))\}$, where $\boldsymbol{\phi}_e$ is a feature map with the associated Mercer kernel $k_e$. Therefore, we can represent nonlinear regression from $\bar{\boldsymbol{e}}(t)$ to $\boldsymbol{e}(t)$ as $A_{\bar{e}} \boldsymbol{\phi}_e(\bar{\boldsymbol{e}}(t))$.

[2] This is given by $(L_*^{-1})' L_*^{-1} = \Sigma_{\phi_* \phi_* | \phi_u}^{-1} \approx (A_*' \hat{\Sigma}_{**|u} A_*)^{-1} = \bar{A}_*' \hat{\Sigma}_{**|u}^{-1} \bar{A}_* = \bar{A}_*' (\hat{L}_*^{-1})' \hat{L}_*^{-1} \bar{A}_*.$
